# FINANCIAL APPLICATIONS OF LEARNING FROM HINTS

**Yaser S. Abu-Mostafa**
California Institute of Technology
and
NeuroDollars, Inc.
e-mail: yaser@caltech.edu

## Abstract

The basic paradigm for learning in neural networks is 'learning from examples' where a training set of input-output examples is used to teach the network the target function. Learning from hints is a generalization of learning from examples where additional information about the target function can be incorporated in the same learning process. Such information can come from common sense rules or special expertise. In financial market applications where the training data is very noisy, the use of such hints can have a decisive advantage. We demonstrate the use of hints in foreign-exchange trading of the U.S. Dollar versus the British Pound, the German Mark, the Japanese Yen, and the Swiss Franc, over a period of 32 months. We explain the general method of learning from hints and how it can be applied to other markets. The learning model for this method is not restricted to neural networks.

## 1  INTRODUCTION

When a neural network learns its target function from examples (training data), it knows nothing about the function except what it sees in the data. In financial market applications, it is typical to have limited amount of relevant training data, with high noise levels in the data. The information content of such data is modest, and while the learning process can try to make the most of what it has, it cannot create new information on its own. This poses a fundamental limitation on the

learning approach, not only for neural networks, but for all other models as well. It is not uncommon to see simple rules such as the moving average outperforming an elaborate learning-from-examples system.

Learning from hints (Abu-Mostafa, 1990, 1993) is a value-added feature to learning from examples that boosts the information content in the data. The method allows us to use prior knowledge about the target function, that comes from common sense or expertise, along with the training data in the *same* learning process. Different types of hints that may be available in a given application can be used simultaneously. In this paper, we give experimental evidence of the impact of hints on learning performance, and explain the method in some detail to enable the readers to try their own hints in different markets.

Even simple hints can result in significant improvement in the learning performance. Figure 1 shows the learning performance for foreign exchange (FX) trading with and without the symmetry hint (see section 3), using only the closing price history. The plots are the Annualized Percentage Returns (cumulative daily, unleveraged, transaction cost included), for a sliding one-year test window in the period from April 1988 to November 1990, averaged over the four major FX markets with more than 150 runs per currency. The error bar in the upper left corner is 3 standard deviations long (based on 253 trading days, assuming independence between different runs). The plots establish a statistically significant differential in performance due to the use of hints. This differential holds for all four currencies.

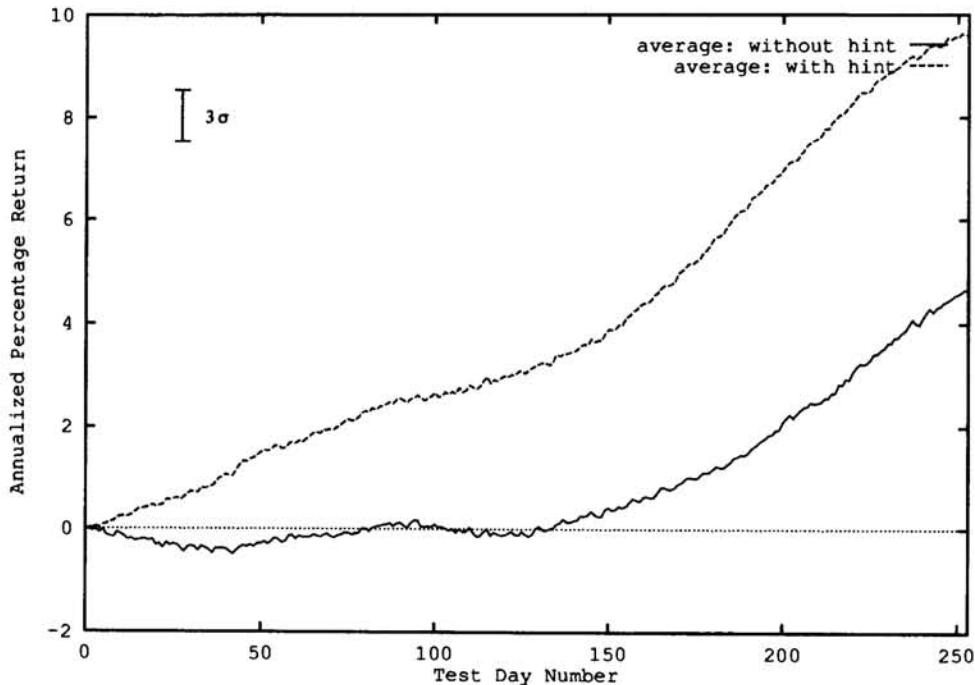

Figure 1: Learning performance with and without hint

Since the goal of hints is to add information to the training data, the differential in performance is likely to be less dramatic if we start out with more informative training data. Similarly, an additional hint may not have a pronounced effect if

we have already used a few hints in the same application. There is a saturation in performance in any market that reflects how well the future can be forecast from the past. (Believers in the Efficient Market Hypothesis consider this saturation to be at zero performance). Hints will not make us forecast a market better than whatever that saturation level may be. They will, however, enable us to approach that level *through learning*.

This paper is organized as follows. Section 2 characterizes the notion of very noisy data by defining the '50% performance range'. We argue that the need for extra information in financial market applications is more pronounced than in other pattern recognition applications. In section 3, we discuss our method for learning from hints. We give examples of different types of hints, and explain how to represent hints to the learning process. Section 4 gives result details on the use of the symmetry hint in the four major FX markets. Section 5 provides experimental evidence that it is indeed the information content of the hint, rather than the incidental regularization effect, that results in the performance differential that we observe.

## 2 FINANCIAL DATA

This section provides a characterization of very noisy data that applies to the financial markets. For a broad treatment of neural-network applications to the financial markets, the reader is referred to (Abu-Mostafa et al, 1994).

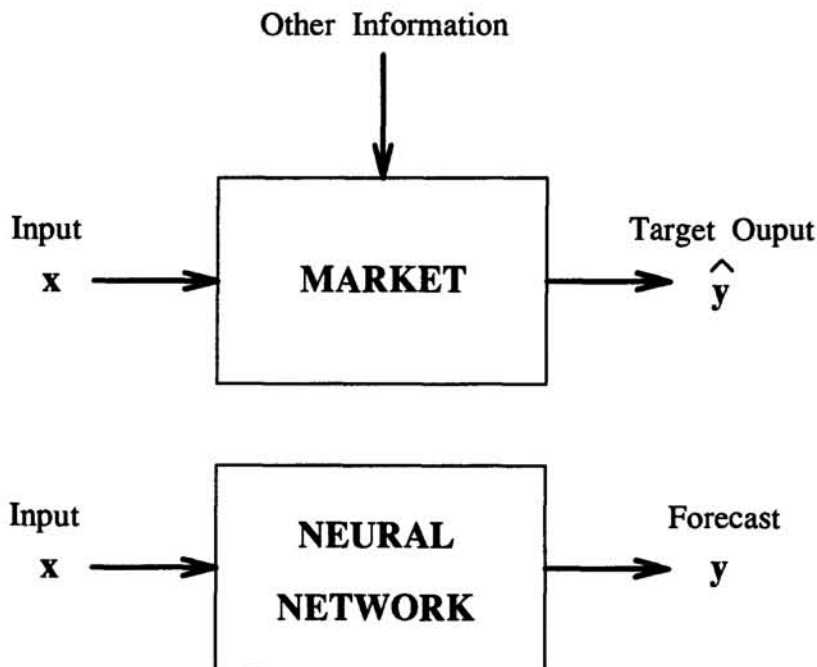

Figure 2: Illustration of the nature of noise in financial markets

Consider the market as a system that takes in a lot of information (fundamentals, news events, rumors, who bought what when, etc.) and produces an output $\hat{y}$ (say up/down price movement for simplicity). A model, e.g., a neural network, attempts

to simulate the market (figure 2), but it takes an input $x$ which is only a small subset of the information. The 'other information' cannot be modeled and plays the role of noise as far as $x$ is concerned. The network cannot determine the target output $\hat{y}$ based on $x$ alone, so it approximates it with its output $y$. It is typical that this approximation will be correct only slightly more than half the time.

What makes us consider $x$ 'very noisy' is that $y$ and $\hat{y}$ agree only $\frac{1}{2} + \epsilon$ of the time (50% performance range). This is in contrast to the typical pattern recognition application, such as optical character recognition, where $y$ and $\hat{y}$ agree $1 - \epsilon$ of the time (100% performance range). It is not the poor performance *per se* that poses a problem in the 50% range, but rather the additional difficulty of learning in this range. Here is why.

In the 50% range, a performance of $\frac{1}{2} + \epsilon$ is good, while a performance of $\frac{1}{2} - \epsilon$ is disastrous. During learning, we need to distinguish between good and bad hypotheses based on a limited set of $N$ examples. The problem with the 50% range is that the number of bad hypotheses that look good on $N$ points is huge. This is in contrast to the 100% range where a good performance is as high as $1 - \epsilon$. The number of bad hypotheses that look good here is limited. Therefore, one can have much more confidence in a hypothesis that was learned in the 100% range than one learned in the 50% range. It is not uncommon to see a random trading policy making good money for a few weeks, but it is very unlikely that a random character recognition system will read a paragraph correctly.

Of course this problem would diminish if we used a very large set of examples, because the law of large numbers would make it less and less likely that $y$ and $\hat{y}$ can agree $\frac{1}{2} + \epsilon$ of the time just by 'coincidence'. However, financial data has the other problem of non-stationarity. Because of the continuous evolution in the markets, old data may represent patterns of behavior that no longer hold. Thus, the relevant data for training purposes is limited to fairly recent times. Put together, noise and non-stationarity mean that the training data will not contain enough information for the network to learn the function. More information is needed, and hints can be the means of providing it.

## 3   HINTS

In this section, we give examples of different types of hints and discuss how to represent them to the learning process. We describe a simple way to use hints that allows the reader to try the method with minimal effort. For a more detailed treatment, please see (Abu-Mostafa, 1993).

As far as our method is concerned, a hint is any property that the target function is known to have. For instance, consider the symmetry hint in FX markets as it applies to the U.S. Dollar versus the German Mark (figure 3). This simple hint asserts that if a pattern in the price history implies a certain move in the market, then this implication holds whether you are looking at the market from the U.S. Dollar viewpoint or the German Mark viewpoint. Formally, in terms of normalized prices, the hint translates to invariance under inversion of these prices.

Is the symmetry hint valid? The ultimate test for this is how the learning performance is affected by the introduction of the hint. The formulation of hints is an art.

We use our experience, common sense, and analysis of the market to come up with a list of what we believe to be valid properties of this market. We then represent these hints in a canonical form as we will see shortly, and proceed to incorporate them in the learning process. The improvement in performance will only be as good as the hints we put in.

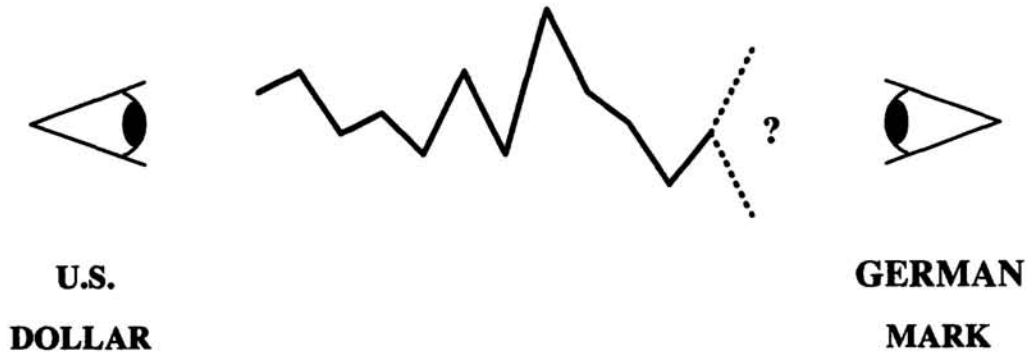

U.S.                                    GERMAN

DOLLAR                                    MARK

Figure 3: Illustration of the symmetry hint in FX markets

The canonical representation of hints is a more systematic task. The first step in representing a hint is to choose a way of generating 'virtual examples' of the hint. For illustration, suppose that the hint asserts that the target function $\hat{y}$ is an odd function of the input. An example of this hint would have the form $\hat{y}(-x) = -\hat{y}(x)$ for a particular input $x$. One can generate as many virtual examples as needed by picking different inputs.

After a hint is represented by virtual examples, it is ready to be incorporated in the learning process along with the examples of the target function itself. Notice that an example of the function is learned by minimizing an error measure, say $(y(x) - \hat{y}(x))^2$, as a way of ultimately enforcing the condition $y(x) = \hat{y}(x)$. In the same way, a virtual example of the oddness hint can be learned by minimizing $(y(x) + y(-x))^2$ as a way of ultimately enforcing the condition $y(-x) = -y(x)$. This involves inputting both $x$ and $-x$ to the network and minimizing the difference between the two outputs. It is easy to show that this can be done using backpropagation (Rumelhart et al, 1986) twice.

The generation of a virtual example of the hint does not require knowing the value of the target function; neither $\hat{y}(x)$ nor $\hat{y}(-x)$ is needed to compute the error for the oddness hint. In fact, $x$ and $-x$ can be artificial inputs. The fact that we do not need the value of the target function is crucial, since it was the limited resource of examples for which we know the value of the target function that got us interested in hints in the first place. On the other hand, for some hints, we can take the examples of the target function that we have, and employ the hint to duplicate these examples. For instance, an example $\hat{y}(x) = 1$ can be used to infer a second example $\hat{y}(-x) = -1$ using the oddness hint. Representing the hint by duplicate examples is an easy way to try simple hints using the same software that we use for learning from examples.

Let us illustrate how to represent two common types of hints. Perhaps the most common type is *the invariance hint*. This hint asserts that $\hat{y}(x) = \hat{y}(x')$ for certain pairs $x, x'$. For instance, "$\hat{y}$ is shift-invariant" is formalized by the pairs $x, x'$ that are shifted versions of each other. To represent the invariance hint, an invariant pair $(x, x')$ is picked as a virtual example. The error associated with this example is $(y(x) - y(x'))^2$. Another related type of hint is *the monotonicity hint*. The hint asserts for certain pairs $x, x'$ that $\hat{y}(x) \leq \hat{y}(x')$. For instance, "$\hat{y}$ is monotonically nondecreasing in $x$" is formalized by the pairs $x, x'$ such that $x < x'$. One application where the monotonicity hint occurs is the extension of personal credit. If person A is identical to person B except that A makes less money than B, then the approved credit line for A cannot exceed that of B. To represent the monotonicity hint, a monotonic pair $(x, x')$ is picked as a virtual example. The error associated with this example is given by $(y(x) - y(x'))^2$ if $y(x) > y(x')$ and zero if $y(x) \leq y(x')$.

## 4  FX TRADING

We applied the symmetry hint in the four FX markets of the U.S. Dollar versus the British Pound, the German Mark, the Japanese Yen, and the Swiss Franc. In each case, only the closing prices for the preceding 21 days were used for inputs. The objective (fitness) function we chose was the total return on the training set, and we used simple filtering methods on the inputs and outputs of the networks. In each run, the training set consisted of 500 days, and the test was done on the following 253 days.

All four currencies show an improved performance when the symmetry hint is used. Roughly speaking, we are in the market half the time, each trade takes 4 days, the hit rate is close to 50%, and the A.P.R. without hint is 5% and with hint is 10% (the returns are annualized, unleveraged, and include the transaction cost; spread and average slippage). Notice that having the return as the objective function resulted in a fairly good return with a modest hit rate.

## 5  CROSS CHECKS

In this final section, we report more experimental results aimed at validating our claim that the information content of the hint is the reason behind the improved performance. Why is this debatable? A hint plays an incidental role as a constraint on the neural network during learning, since it restricts the solutions the network may settle in. Because overfitting is a common problem in learning from examples, any restriction may improve the out-of-sample performance by reducing overfitting (Akaike, 1969, Moody, 1992). This is the idea behind regularization.

To isolate the informative role from the regularizing role of the symmetry hint, we ran two experiments. In the first experiment, we used an uninformative hint, or 'noise' hint, which provides a random target output for the same inputs used in the examples of the symmetry hint. Figure 4 contrasts the performance of the noise hint with that of the real symmetry hint, averaged over the four currencies. Notice that the performance with the noise hint is close to that without any hint (figure 1), which is consistent with the notion of uninformative hint. The regularization effect seems to be negligible.

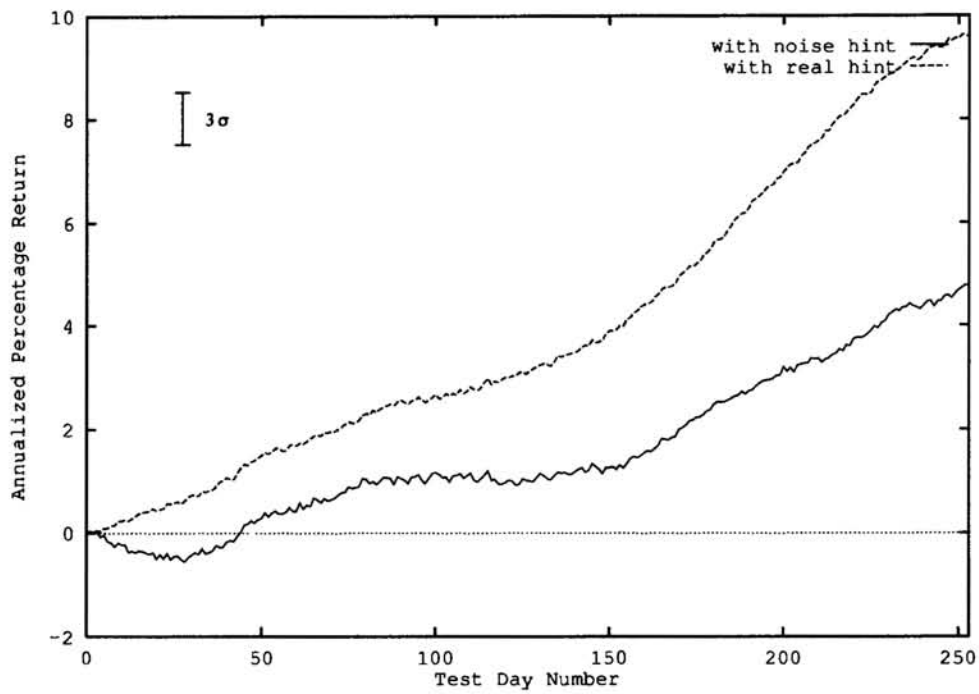

Figure 4: Performance of the real hint versus a noise hint

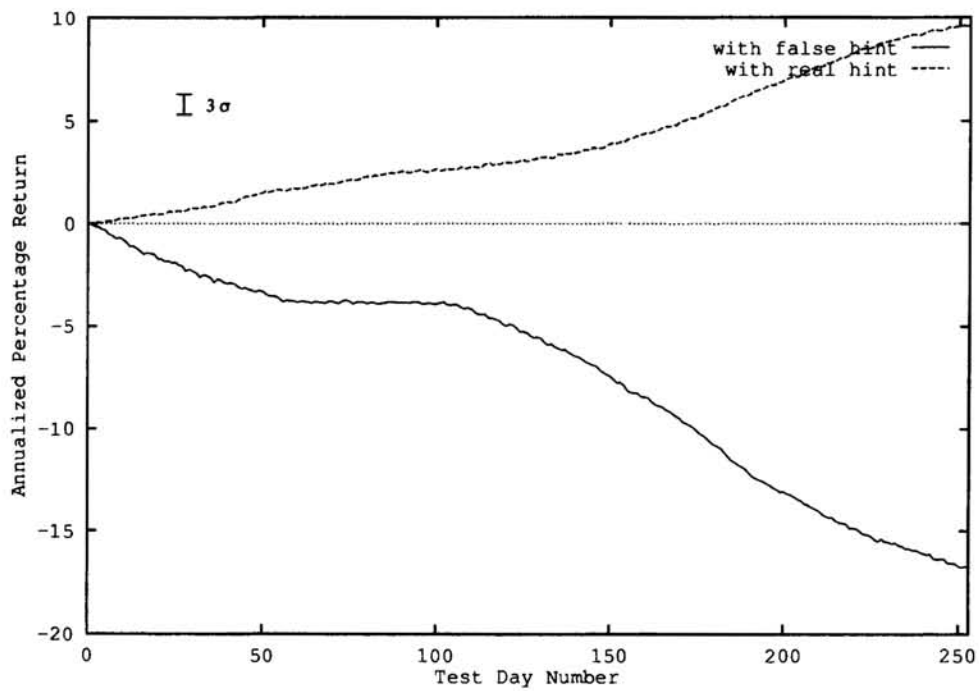

Figure 5: Performance of the real hint versus a false hint

In the second experiment, we used a harmful hint, or 'false' hint, in place of the symmetry hint. The hint takes the same examples used in the symmetry hint and asserts antisymmetry instead. Figure 5 contrasts the performance of the false hint with that of the real symmetry hint. As we see, the false hint had a detrimental effect on the performance. This is consistent with the hypothesis that the symmetry hint is valid, since its negation results in worse performance than no hint at all. Notice that the transaction cost is taken into consideration in all of these plots, which works as a negative bias and amplifies the losses of bad trading policies.

## CONCLUSION

We have explained learning from hints, a systematic method for combining rules and data in the same learning process, and reported experimental results of a statistically significant improvement in performance in the four major FX markets that resulted from using a simple symmetry hint. We have described different types of hints and simple ways of using them in learning, to enable the readers to try their own hints in different markets.

### Acknowledgements

I would like to acknowledge Dr. Amir Atiya for his valuable input. I am grateful to Dr. Ayman Abu-Mostafa for his expert remarks.

### References

Abu-Mostafa, Y. S. (1990), Learning from hints in neural networks, *Journal of Complexity* **6**, pp. 192-198.

Abu-Mostafa, Y. S. (1993), A method for learning from hints, *Advances in Neural Information Processing Systems* **5**, S. Hanson et al (eds), pp. 73-80, Morgan-Kaufmann.

Abu-Mostafa, Y. S. et al (eds) (1994), *Proceedings of Neural Networks in the Capital Markets*, Pasadena, California, November 1994.

Akaike, H. (1969), Fitting autoregressive models for prediction, *Ann. Inst. Stat. Math.* **21**, pp. 243-247.

Moody, J. (1992), The effective number of parameters: An analysis of generalization and regularization in nonlinear learning systems, in *Advances in Neural Information Processing Systems* **4**, J. Moody et al (eds), pp. 847-854, Morgan Kaufmann.

Rumelhart, D. E., Hinton, G. E., and Williams, R. J. (1986), Learning internal representations by error propagation, *Parallel Distributed Processing* **1**, D. Rumelhart et al, pp. 318-362, MIT Press.

Weigend, A., Rumelhart, D., and Huberman, B. (1991), Generalization by weight elimination with application to forecasting, in *Advances in Neural Information Processing Systems* **3**, R. Lippmann et al (eds), pp. 875-882, Morgan Kaufmann.